# Understanding Brain Connectivity Patterns during Motor Imagery for Brain-Computer Interfacing

**Moritz Grosse-Wentrup**
Max Planck Institute for Biological Cybernetics
Spemannstr. 38
72076 Tübingen, Germany
moritzgw@ieee.org

## Abstract

EEG connectivity measures could provide a new type of feature space for inferring a subject's intention in Brain-Computer Interfaces (BCIs). However, very little is known on EEG connectivity patterns for BCIs. In this study, EEG connectivity during motor imagery (MI) of the left and right is investigated in a broad frequency range across the whole scalp by combining Beamforming with Transfer Entropy and taking into account possible volume conduction effects. Observed connectivity patterns indicate that modulation intentionally induced by MI is strongest in the $\gamma$-band, i.e., above 35 Hz. Furthermore, modulation between MI and rest is found to be more pronounced than between MI of different hands. This is in contrast to results on MI obtained with bandpower features, and might provide an explanation for the so far only moderate success of connectivity features in BCIs. It is concluded that future studies on connectivity based BCIs should focus on high frequency bands and consider experimental paradigms that maximally vary cognitive demands between conditions.

## 1 Introduction

Brain-Computer Interfaces (BCIs) are devices that enable a subject to communicate without utilizing the peripheral nervous system, i.e., without any overt movement requiring volitional motor control. The primary goal of research on BCIs is to enable basic communication for subjects unable to communicate by normal means due to neuro-degenerative diseases such as amyotrophic lateral sclerosis (ALS). In non-invasive BCIs, this is usually approached by measuring the electric field of the brain by EEG, and detecting changes intentionally induced by the subject (cf. [1] for a general introduction to BCIs). The most commonly used experimental paradigm in this context is motor imagery (MI) [2]. In MI subjects are asked to haptically imagine movements of certain limbs, e.g., the left or the right hand. MI is known to be accompanied by a decrease in bandpower (usually most prominent in the $\mu$-band, i.e., roughly at 8-13 Hz) in that part of the motor cortex representing the specific limb [3]. These bandpower changes, termed event related (de-)synchronization (ERD/ERS), can be detected and subsequently used for inferring the subject's intention. This approach to BCIs has been demonstrated to be very effective in healthy subjects, with only little subject training time required to achieve classification accuracies close to 100% in two-class paradigms [4–6]. Furthermore, satisfactory classification results have been reported with subjects in early to middle stages of ALS [7]. However, all subjects diagnosed with ALS and capable of operating a BCI still had residual motor control that enabled them to communicate without the use of a BCI. Until now, no communication has been established with a completely locked-in subject, i.e., a subject without any residual motor control. Establishing communication with a completely locked-in subject arguably constitutes the most important challenge in research on BCIs.

Unfortunately, reasons for the failure of establishing communication with completely locked-in subjects remain unknown. While cognitive deficits in completely locked-in patients can at present not be ruled out as the cause of this failure, another possible explanation is abnormal brain activity observed in patients in late stages of ALS [8]. Our own observations indicate that intentionally induced bandpower changes in the electric field of the brain might be reduced in subjects in late stages of ALS. To explore the plausibility of this explanation for the failure of current BCIs in completely locked-in subjects, it is necessary to devise feature extraction algorithms that do not rely on measures of bandpower. In this context, one promising approach is to employ connectivity measures between different brain regions. It is well known from fMRI-studies that brain activity during MI is not confined to primary motor areas, but rather includes a distributed network including pre-motor, parietal and frontal regions of the brain [9]. Furthermore, synchronization between different brain regions is known to be an essential feature of cognitive processing in general [10]. Subsequently, it can be expected that different cognitive tasks, such as MI of different limbs, are associated with different connectivity patterns between brain regions. These connectivity patterns should be detectable from EEG recordings, and thus offer a new type of feature space for inferring a subject's intention. Since measures of connectivity are, at least in principle, independent of bandpower changes, this might offer a new approach to establishing communication with completely locked-in subjects.

In recent years, several measures of connectivity have been developed for analyzing EEG recordings (cf. [11] for a good introduction and a comparison of several algorithms). However, very few studies exist that analyze connectivity patterns as revealed by EEG during MI [12, 13]. Furthermore, these studies focus on differences in connectivity patterns between MI and motor execution, which is not of primary interest for research on BCIs. In the context of non-invasive BCIs, connectivity measures have been most notably explored in [14] and [15]. However, these studies only consider frequency bands and small subsets of electrodes known to be relevant for bandpower features, and do not take into account possible volume conduction effects. This might lead to misinterpreting bandpower changes as changes in connectivity. Consequently, a better understanding of connectivity patterns during MI of different limbs as measured by EEG is required to guide the design of new feature extraction algorithms for BCIs. Specifically, it is important to properly address possible volume conduction effects, not confine the analysis to a small subset of electrodes, and consider a broad range of frequency bands.

In this work, these issues are addressed by combining connectivity analysis during MI of the left and right hand in four healthy subjects with Beamforming methods [6]. Since it is well known that MI includes primary motor cortex [3], this area is chosen as the starting point of the connectivity analysis. Spatial filters are designed that selectively extract those components of the EEG originating in the left and right motor cortex. Then, the concept of Transfer Entropy [16] is used to estimate class-conditional 'information flow' from all 128 employed recording sites into the left and right motor cortex in frequency bands ranging from 5 - 55 Hz. In this way, spatial topographies are obtained for each frequency band that depict by how much each area of the brain is influencing the left/right motor cortex during MI of the left/right hand. Interestingly, the most pronounced changes in connectivity patterns are not observed in MI of the left vs. the right hand, but rather in rest vs. MI of either hand. Furthermore, these pattern changes are most pronounced in frequency bands not usually associated with MI. i.e., in the $\gamma$-band above 35 Hz. These results suggest that in order to fully exploit the capabilities of connectivity measures for BCIs, and establish communication with completely locked-in subjects, it might be advisable to consider $\gamma$-band oscillations and adapt experimental paradigms as to maximally vary cognitive demands between conditions.

## 2 Methods

### 2.1 Symmetric vs. Asymmetric Connectivity Analysis

In analyzing interrelations between time-series data it is important to distinguish symmetric from asymmetric measures. Consider Fig. 1, depicting two graphs of three random processes $s_1$ to $s_3$, representing three EEG sources. The goal of symmetric connectivity analysis (Fig. 1.a) is to estimate some instantaneous measure of similarity between random processes, i.e., assigning weights to the undirected edges between the nodes of the graph in Fig. 1.a. Amplitude coupling and phase synchronization fall into this category, which are the measures employed in [14] and [15] for feature extraction in BCIs. However, interrelations between EEG sources originating in different regions of

a)                                                        b)

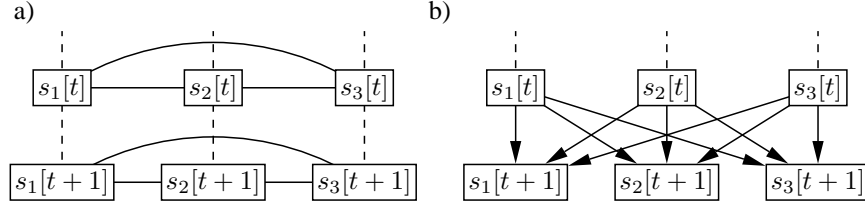

Figure 1: Illustration of symmetric- vs. asymmetric connectivity analysis for three EEG sources within the brain.

the brain can be expected to be asymmetric, with certain brain regions exerting stronger influence on other regions than vice versa. For this reason, asymmetric connectivity measures potentially provide more information on cognitive processes than symmetric measures.

Considering asymmetric relations between random processes requires a definition of how the influence of one process on another process is to be measured, i.e., a quantitative definition of causal influence. The commonly adopted definition of causality in time-series analysis is that $s_i$ causes $s_j$ if observing $s_i$ helps in predicting future observations of $s_j$, i.e., reduces the prediction error of $s_j$. This implies that cause precedes effect, i.e., that the graph in Fig. 1.b may only contain directed arrows pointing forward in time. Note that there is some ambiguity in this definition of causality, since it does not specify a metric for reduction of the prediction error of $s_j$ due to observing $s_i$. In Granger causality (cf. [11]), reduction of the variance of the prediction error is chosen as a metric, essentially limiting Granger causality to linear systems. It should be noted, however, that any other metric is equally valid. Finally, note that for reasons of simplicity the graph in Fig. 1.b only contains directed edges from nodes at time $t$ to nodes at time $t+1$. In general, directed arrows from nodes at times $t, \ldots, t-k$ to nodes at time $t+1$ may be considered, with $k$ the order of the random processes generating $\boldsymbol{s}[t+1]$.

To assess Granger causality between bivariate time-series data a linear autoregressive model is fit to the data, which is then used to compute a 2x2 transfer matrix in the frequency domain (cf. [11]). The off-diagonal elements of the transfer matrix then provide a measure of the asymmetric interaction between the observed time-series. Extensions of Granger causality to multivariate time-series data, termed directed transfer function (DTF) and partial directed coherence (PDC), have been developed (cf. [11] and the references therein). However, in this work a related but different measure for asymmetric interrelations between time-series is utilized. The concept of Transfer Entropy (TE) [16] defines the causal influence of $s_i$ on $s_j$ as the reduction in entropy of $s_j$ obtained by observing $s_i$. More precisely, let $s_i$ and $s_j$ denote two random processes, and let $\boldsymbol{s}_{i/j}^k[t] := \big(s_{i/j}[t], \ldots, s_{i/j}[t-k]\big)$. TE is then defined as

$$T_k\left(s_i[t] \to s_j[t+1]\right) := H\left(s_j[t+1]|\boldsymbol{s}_j^k[t]\right) - H\left(s_j[t+1]|\boldsymbol{s}_j^k[t], \boldsymbol{s}_i^k[t]\right), \qquad (1)$$

with $k$ the order of the random processes and $H(\cdot)$ the Shannon entropy. TE can thus be understood as the reduction in uncertainty about the random process $s_j$ at time $t+1$ due to observing the past $k$ samples of the random process $s_i$. Both, Granger causality and TE, thus define causal influence as a reduction in the uncertainty of a process due to observing another process, but employ different metrics to measure reduction in uncertainty. While TE is a measure that applies to any type of random processes, it is difficult to compute in practice. Hence, in this study only Gaussian processes are considered, i.e., it is assumed that $\big(s_j[t+1], \boldsymbol{s}_j^k[t], \boldsymbol{s}_i^k[t]\big)$ is jointly Gaussian distributed. TE can then be computed as

$$T_k^{\mathrm{GP}}\left(s_i[t] \to s_j[t+1]\right) = \frac{1}{2}\log \frac{\det R_{(\boldsymbol{s}_j^k[t], \boldsymbol{s}_i^k[t])} \det R_{(s_j[t+1], \boldsymbol{s}_j^k[t])}}{\det R_{(s_j[t+1], \boldsymbol{s}_j^k[t], \boldsymbol{s}_i^k[t])} \det R_{(\boldsymbol{s}_j^k[t])}}, \qquad (2)$$

with $R_{(\cdot)}$ the (cross-)covariance matrices of the respective random processes [17]. In comparison to Granger causality and related measures, TE for Gaussian processes possesses several advantages. It is easy to compute from a numerical perspective, since it does not require fitting a multivariate autoregressive model including (implicit) inversion of large matrices. Furthermore, for continuous processes it is invariant under coordinate transformations [17]. Importantly, this entails invariance with regard to scaling of the random processes.

Computing TE for Gaussian processes requires estimation of the (cross-)covariance matrices in (2). Consider a matrix $S \in \mathbb{R}^{2 \times T \times N}$, corresponding to data recorded from two EEG

sources during an experimental paradigm with $N$ trials of $T$ samples each. In order to compute $T_k^{GP}(s_1[t] \rightarrow s_2[t+1])$ for $t = k+1, \ldots, T-k-1$, it is assumed that in each trial $s_1[t]$ and $s_2[t]$ are i.i.d. samples from the distribution $p(s_1[t], s_2[t])$, i.e., that the non-stationary Gaussian processes that give rise to the observation matrix $S$ are identical for each of the $N$ repetitions of the experimental paradigm. For each instant in time, TE can then be evaluated by computing the sample (cross-)covariance matrices required in (2) across trials. Note that evaluating (2) requires specification of $k$. In general, $k$ should be chosen as large as possible in order to maximize information on the random processes contained in the (cross-)covariance matrices. However, choosing $k$ too large leads to rank deficient matrices with a determinant of zero. Here, for each observation matrix $S$ the highest possible $k$ is chosen such that none of the matrices in (2) is rank deficient.

## 2.2 The Problem of Volume Conduction in EEG Connectivity Analysis

The goal of connectivity analysis in EEG recordings is to estimate connectivity patterns between different regions of the brain. Unfortunately, EEG recordings do not offer direct access to EEG sources. Instead, each EEG electrode measures a linear and instantaneous superposition of EEG sources within the brain [18]. This poses a problem for symmetric connectivity measures, since these assess instantaneous coupling between electrodes [18]. Asymmetric connectivity measures such as TE, on the other hand, are not based on instantaneous coupling, but rather consider prediction errors. It is not obvious that instantaneous volume conduction also poses a problem for this type of measures. Unfortunately, the following example demonstrates that volume conduction also leads to incorrect connectivity estimates in asymmetric connectivity analysis based on TE.

**Example 1 (Volume Conduction Effects in Connectivity Analysis based on Transfer Entropy)**
*Consider the EEG signals $x_1[t]$ and $x_2[t]$, recorded at two electrodes placed on the scalp, that consist of a linear superposition of three EEG sources $s_1[t]$ to $s_3[t]$ situated somewhere within the brain (Fig. 2.a). Let $\boldsymbol{x}[t] = (x_1[t], \; x_2[t])^T$ and $\boldsymbol{s}[t] = (s_1[t], \; s_2[t], \; s_3[t])^T$. Then $\boldsymbol{x}[t] = A\boldsymbol{s}[t]$, with $A \in \mathbb{R}^{2 \times 3}$ describing the projection strength of each source to each electrode. For sake of simplicity, assume that $A = (1 \; 0 \; 1 \; ; \; 0 \; 1 \; 1)$, i.e., that the first source only projects to the first electrode with unit strength, the second source only projects to the second electrode with unit strength, and the third source projects to both electrodes with unit strength. Furthermore, assume that*

$$p(\boldsymbol{s}[t+1], \boldsymbol{s}[t]) = \mathcal{N}(\mathbf{0}, \Sigma) \text{ with } \Sigma = \begin{bmatrix} 1 & 0 & 0 & 0 & 0 & 0 \\ 0 & 1 & 0 & 0 & 0 & 0 \\ 0 & 0 & 1 & 0 & 0 & \gamma \\ 0 & 0 & 0 & 1 & 0 & 0 \\ 0 & 0 & 0 & 0 & 1 & 0 \\ 0 & 0 & \gamma & 0 & 0 & 1 \end{bmatrix}, \tag{3}$$

*i.e., that all sources have zero mean, unit variance, are mutually independent, and $s_1$ and $s_2$ are uncorrelated in time. Only $s_3[t]$ and $s_3[t+1]$ are assumed to be correlated with covariance $\gamma$ (Fig. 2.b). In this setting, it would be desirable to obtain zero TE between both electrodes, since there is no interaction between the sources giving rise to the EEG. However, some rather tedious algebraic manipulations reveal that in this case*

$$T_1^{GP}(x_2[t] \rightarrow x_1[t+1]) = \frac{1}{2}\log\left(\frac{3}{2}\right) + \frac{1}{2}\log\left(\frac{4-\gamma^2}{6-2\gamma^2}\right). \tag{4}$$

*Note that (4) is zero if and only if $\gamma = 0$, i.e., if $s_3$ represents white noise. Otherwise, TE between the two electrodes is estimated to be greater than zero solely due to volume conduction effects from source $s_3$. Further note that qualitatively this result holds independently of the strength of the projection of the third source to both electrodes.*

## 2.3 Attenuation of Volume Conduction Effects via Beamforming

One way to avoid volume conduction effects in EEG connectivity analysis is to perform source localization on the obtained EEG data, and apply connectivity measures on estimated current density time-series at certain locations within the brain [11]. This is feasible to test certain hypothesis, e.g., to evaluate whether there exists a causal link between two specific points within the brain. However, testing pairwise causal links between more than just a few points within the brain is computationally

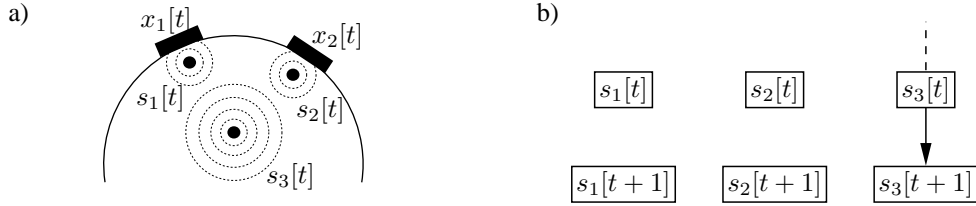

Figure 2: Illustration of volume conduction effects in EEG connectivity analysis.

intractable. Accordingly, attenuation of volume conduction effects via source localization is not feasible if a complete connectivity pattern considering the whole brain is desired. Here, a different approach is pursued. It is well known that primary motor cortex is central to MI as measured by EEG [3]. Accordingly, it is assumed that any brain region involved in MI displays some connectivity to the primary motor cortex. This (admittedly rather strong) assumption enables a complete analysis of the connectivity patterns during MI covering the whole brain in the following way. First, two spatial filters, commonly known as Beamformers, are designed that selectively extract EEG sources originating within the right and left motor cortex, respectively [6]. In brief, this can be accomplished by solving the optimization problem

$$\boldsymbol{w}^* = \underset{\boldsymbol{w} \in \mathbb{R}^M}{\operatorname{argmax}} \left\{ \frac{\boldsymbol{w}^{\mathrm{T}} R_{\tilde{\boldsymbol{x}}_{l/r}} \boldsymbol{w}}{\boldsymbol{w}^{\mathrm{T}} R_{\boldsymbol{x}} \boldsymbol{w}} \right\}, \tag{5}$$

with $R_{\boldsymbol{x}} \in \mathbb{R}^{M \times M}$ the covariance of the recorded EEG, and $R_{\tilde{\boldsymbol{x}}_{l/r}} \in \mathbb{R}^{M \times M}$ model-based spatial covariance matrices of EEG sources originating within the left/right motor cortex. In this way, spatial filters can be obtained that optimally attenuate the variance of all EEG sources not originating within the left/right motor cortex. The desired spatial filters are obtained as the eigenvectors with the largest eigenvalue of the generalized eigenvalue problem $R_{\tilde{\boldsymbol{x}}_{l/r}} \boldsymbol{w} = \lambda R_{\boldsymbol{x}} \boldsymbol{w}$ (cf. [6] for a more detailed presentation).

With EEG sources originating within the left and right motor cortex extracted, TE from all EEG electrodes into the left and right motor cortex can be computed. In this way, volume conduction effects from all sources within the brain into the left/right motor cortex can be optimally attenuated. However, volume conduction effects from the left/right motor cortex to any of the EEG electrodes still poses a problem. Accordingly, it has to be verified if any positive TE from an EEG electrode into the left/right motor cortex could be caused by bandpower changes within the left/right motor cortex. Positive TE from any electrode into the left/right motor cortex can only be considered as a genuine causal link if it is not accompanied by a bandpower change in the respective motor cortex.

## 3 Experimental Results

To investigate connectivity patterns during MI the following experimental paradigm was employed. Subjects sat in a dimly lit and shielded room, approximately two meters in front of a silver screen. Each trial started with a centrally displayed fixation cross. After three seconds, the fixation cross was overlaid with a centrally placed arrow pointing to the left or right. This instructed subjects to begin MI of the left or right hand, respectively. Subjects were explicitly instructed to perform haptic MI, but the exact choice of the type of imaginary hand movement was left unspecified. After a further seven seconds the arrow was removed, indicating the end of the trial and start of the next trial. 150 trials per class were carried out by each subjects in randomized order. During the experiment, EEG was recorded at 128 electrodes placed according to the extended 10-20 system with electrode Cz as reference. EEG data was re-referenced to common average reference offline. Four healthy subjects participated in the experiment, all of which were male and right handed with an age of $27 \pm 2.5$ years. For each subject, electrode locations were recorded with an ultrasound tracking system. No artifact correction was employed and no trials were rejected.

For each subject, model-based covariance matrices $R_{\tilde{\boldsymbol{x}}_{l/r}}$ for EEG sources within the left/right motor cortex were computed as described in [6]. The EEG covariance matrix $R_{\boldsymbol{x}}$ was computed for each subject using all available data, and the two desired Beamformers, extracting EEG sources from the left and right motor cortex, were computed by solving (5). The EEG sources extracted from the left/right motor cortex as well as the unfiltered data recorded at each electrode were then bandpass-

filtered with sixth-order Butterworth filters in five frequency bands ranging from 5 to 55 Hz in steps of 10 Hz. Then, TE was computed from all EEG electrodes into the left/right motor cortex at each sample point as described in Section 2.1. Furthermore, for each subject class-conditional bandpower changes (ERD/ERS) of sources extracted from the left/right motor cortex were computed in order to identify frequency bands with common modulations in bandpower and TE. Two subjects showed significant modulations of bandpower in all five frequency bands. These were excluded from further analysis, since any observed positive TEs could have been confounded by volume conduction. The resulting topographies of mean TE between conditions of the two remaining subjects are shown in Fig. 3. Here, the first two columns show mean TE from all electrodes into the left/right motor cortex during MI of either hand (3.5-10s) minus mean TE during baseline (0.5-3s) in each of the five frequency bands. The last two columns show mean differences in TE into the left/right motor cortex between MI of the left and right hand (both conditions also baseline corrected). Note that the topographies in Fig. 3 have been normalized to the maximum difference across conditions to emphasize differences between conditions. Interestingly, no distinct differences in TE are observed between MI of the left and right hand. Instead, strongest differences in TE are observed in rest vs. MI of either hand (left two columns). The amount of decrease in TE during MI relative to rest increases with higher frequencies, and is most pronounced in the $\gamma$-band from 45-55 Hz (last row, left two columns). Topographically, strongest differences are observed in frontal, pre-central, and post-central areas. Observed changes in TE are statistically significant with significance level $\alpha = 0.01$ at all electrodes in Fig. 3 marked with red crosses (statistical significance was tested non-parametrically and individually for each subject, Beamformer, and condition by one thousand times randomly permuting the EEG data of each recorded trial in time and testing the null-hypothesis that changes in TE at least as large as those in Fig.3 are observed without any temporal structure being present in the data). Due to computational resources only a small subset of electrodes was tested for significance. The observed changes in TE display opposite modulations in comparison to mean bandpower changes observed in left/right motor cortex relative to baseline (Fig. 4, only significant ($\alpha = 0.01$) bandpower changes relative to baseline (0-3s) plotted). Here, strongest modulation of bandpower is found in the $\mu$- ($\sim 10$ Hz) and $\beta$-band ($\sim 25$ Hz). Frequencies above 35 Hz show very little modulation, indicating that the observed differences in TE at high frequencies in Fig. 3 are not due to volume conduction but genuine causal links.

## 4 Discussion

In this study, Beamforming and TE were employed to investigate the topographies of 'information flow' into the left and right motor cortex during MI as measured by EEG. To the best of the author's knowledge, this is the first study investigating asymmetric connectivity patterns between brain regions during MI of different limbs considering a broad frequency range, a large number of recordings sites, and properly taking into account volume conduction effects. However, it should be pointed out that there are several issues that warrant further investigation. First, the presented results are obtained from only two subjects, since two subjects had to be excluded due to possible volume conduction effects. Future studies with more subjects are required to validate the obtained results. Also, no outflow from primary motor cortex and no TE between brain regions not including primary motor cortex have been considered. Finally, the methodology presented in this study can not be applied in a straight-forward manner to single-trial data, and is thus only of limited use for actual feature extraction in BCIs.

Never the less, the obtained results indicate that bandpower changes in motor cortex and connectivity between motor cortex and other regions of the brain are processes that occupy distinct spectral bands and are modulated by different cognitive tasks. In conjunction with the observation of no distinct changes in connectivity patterns between MI of different limbs, this indicates that in [14] and [15] bandpower changes might have been misinterpreted as connectivity changes. This is further supported by the fact that these studies focused on frequency bands displaying significant modulation of bandpower (8-30 Hz) and did not control for volume conduction effects. In conclusion, the pronounced modulation of connectivity between MI of either hand vs. rest in the $\gamma$-band observed in this study underlines the importance of also considering high frequency bands in EEG connectivity analysis. Furthermore, since the $\gamma$-band is thought to be crucial for dynamic functional connectivity between brain regions [10], future studies on connectivity patterns in BCIs should consider experimental paradigms that maximally vary cognitive demands in order to activate different networks within the brain across conditions.

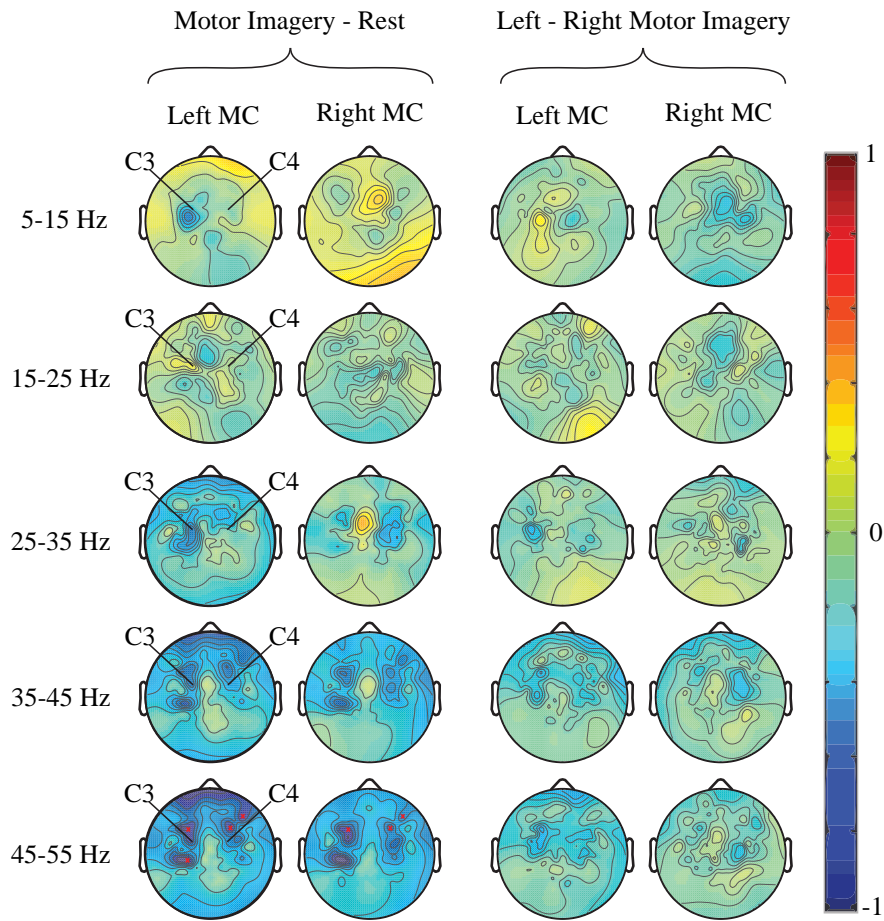

Figure 3: Topographies of mean Transfer Entropy changes into left/right motor cortex (MC). C3/C4 mark electrodes over left/right motor cortex. Red crosses indicate statistically significant electrodes. Plotted with [19].

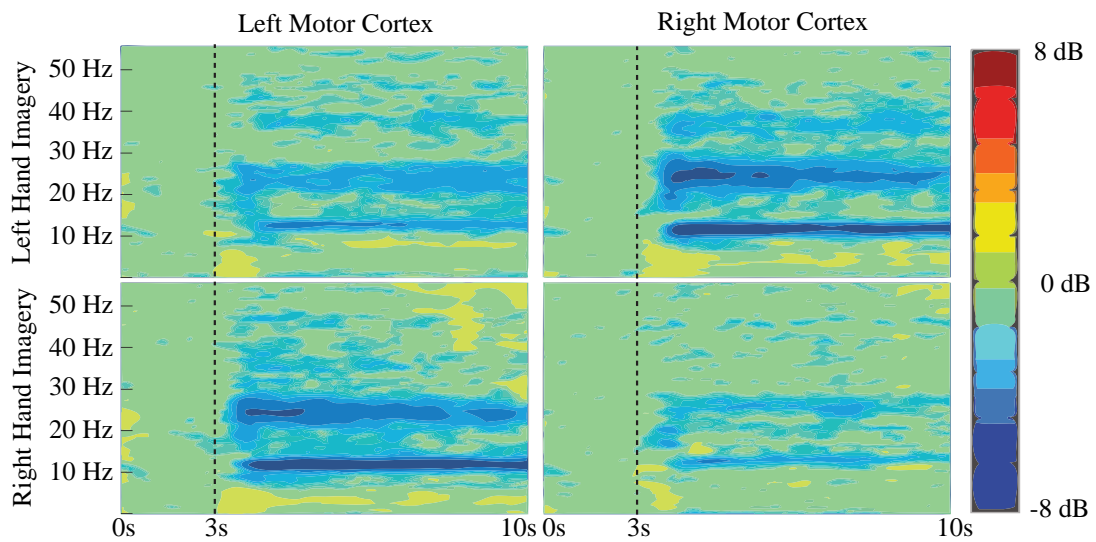

Figure 4: Class-conditional mean ERD/ERS in left/right motor cortex relative to baseline (0-3s). Horizontal line marks start of motor imagery. Plotted with [19].

# References

[1] J.R. Wolpaw, N. Birbaumer, D.J. McFarland, G. Pfurtscheller, and T.M. Vaughan. Brain-computer interfaces for communication and control. Clinical Neurophysiology, 113(6):767–791, 2002.

[2] S.G. Mason, A. Bashashati, M. Fatourechi, K.F. Navarro, and G.E. Birch. A comprehensive survey of brain interface technology designs. Annals of Biomedical Engineering, 35(2):137–169, 2007.

[3] G. Pfurtscheller and F.H. Lopes da Silva. Even-related EEG/MEG synchronization and desynchronization: basic principles. Clinical Neurophysiology, 110:1842–1857, 1999.

[4] H. Ramoser, J. Mueller-Gerking, and G. Pfurtscheller. Optimal spatial filtering of single trial EEG during imagined hand movement. IEEE Transactions on Rehab. Eng., 8(4):441–446, 2000.

[5] B. Blankertz, G. Dornhege, M. Krauledat, K.R. Mueller, and G. Curio. The non-invasive Berlin brain-computer interface: Fast acquisition of effective performance in untrained subjects. NeuroImage, 27(2):539–550, 2007.

[6] Moritz Grosse-Wentrup, Klaus Gramann, and Martin Buss. Adaptive spatial filters with pre-defined region of interest for EEG based brain-computer-interfaces. In B. Schoelkopf, J. Platt, and T. Hoffman, editors, Advances in Neural Information Processing Systems 19, pages 537–544. MIT Press, Cambridge, MA, 2007.

[7] A. Kubler, F. Nijboer, J. Mellinger, T.M. Vaughan, H. Pawelzik, G. Schalk, D.J. McFarland, N. Birbaumer, and J. Wolpaw. Patients with ALS can use sensorimotor rhythms to operate a brain-computer interface. Neurology, 64:1775–1777, 2005.

[8] B. Kotchoubey, S. Lang, S. Winter, and N. Birbaumer. Cognitive processing in completely paralyzed patients with amyotrophic lateral sclerosis. European Journal of Neurology, 10(5):551–558, 2003.

[9] T. Hanakawa, M.A. Dimyan, and M. Hallett. Motor planning, imagery, and execution in the distributed motor network: a time-course study with functional MRI. Cerebral Cortex. Advance online publication.

[10] F. Varela, J.-P. Lachaux, E. Rodriguez, and J. Martinerie. The brainweb: phase sychronization and large-scale integration. Nature Reviews Neuroscience, 2:229–239, 2001.

[11] L. Astolfi, F. Cincotti, D. Mattia, M.G. Marciani, L.A. Baccala, F. de Vico Fallani, S. Salinari, M. Ursino, M. Zavaglia, L. Ding, J.C. Edgar, G.A. Miller, B. He, and F. Babiloni. Comparison of different cortical connectivity estimators for high-resolution EEG recordings. Human Brain Mapping, 28:143–157, 2007.

[12] R. Kus, J.S. Ginter, and K.J. Blinowska. Propagation of EEG activity during finger movement and its imagination. Acta Neurobiologiae Experimentalis, 66:195–206, 2006.

[13] M.L. Stavrinou, L. Moraru, L. Cimponeriu, S. Della Penna, and A. Bezerianos. Evaluation of cortical connectivity during real and imagined rythmic finger tapping. Brain Topography, 19:137–145, 2007.

[14] E. Gysels and P. Celka. Phase synchronization for the recognition of mental tasks in a brain-computer interface. IEEE Transactions on Rehab. Eng., 12(4):406–415, 2004.

[15] Q. Wei, Y. Wang, X. Gao, and S. Gao. Amplitude and phase coupling measures for feature extraction in an EEG-based brain-computer interface. Journal of Neural Engineering, 4:120–129, 2007.

[16] T. Schreiber. Measuring information transfer. Physical Review Letters, 85(2):461–464, 2000.

[17] A. Kaiser and T. Schreiber. Information transfer in continuous processes. Physica D, 166:43–62, 2002.

[18] P.L. Nunez and R. Srinivasan. Electric Fields of the Brain: The Neurophysics of EEG. Oxford University Press, 2005.

[19] A. Delorme and S. Makeig. EEGLAB: an open source toolbox for analysis of single-trial EEG dynamics including independent component analysis. Journal of Neuroscience Methods, 134(1):9–21, 2004.

